# Neuronal Fiber Delineation in Area of Edema from Diffusion Weighted MRI

**Ofer Pasternak**[*]
School of Computer Science
Tel-Aviv University
Tel-Aviv, ISRAEL 69978
oferpas@post.tau.ac.il

**Nir Sochen**
Department of Applied Mathematics
Tel-Aviv University
sochen@post.tau.ac.il

**Nathan Intrator**
School of Computer Science
Tel-Aviv University
nin@post.tau.ac.il

**Yaniv Assaf**
Department of Neurobiochemistry
Faculty of Life Science
Tel-Aviv University
assafyan@post.tau.ac.il

## Abstract

Diffusion Tensor Magnetic Resonance Imaging (DT-MRI) is a non invasive method for brain neuronal fibers delineation. Here we show a modification for DT-MRI that allows delineation of neuronal fibers which are infiltrated by edema. We use the Muliple Tensor Variational (MTV) framework which replaces the diffusion model of DT-MRI with a multiple component model and fits it to the signal attenuation with a variational regularization mechanism. In order to reduce free water contamination we estimate the free water compartment volume fraction in each voxel, remove it, and then calculate the anisotropy of the remaining compartment. The variational framework was applied on data collected with conventional clinical parameters, containing only six diffusion directions. By using the variational framework we were able to overcome the highly ill posed fitting. The results show that we were able to find fibers that were not found by DT-MRI.

## 1 Introduction

Diffusion weighted Magnetic Resonance Imaging (DT-MRI) enables the measurement of the apparent water self-diffusion along a specified direction [1]. Using a series of Diffusion Weighted Images (DWIs) DT-MRI can extract quantitative measures of water molecule diffusion anisotropy which characterize tissue microstructure [2]. Such measures are in particular useful for the segmentation of neuronal fibers from other brain tissue which then allows a noninvasive delineation and visualization of major brain neuronal fiber bundles in vivo [3]. Based on the assumptions that each voxel can be represented by a single diffusion compartment and that the diffusion within this compartment has a Gaussian distribution

---

[*]http://www.cs.tau.ac.il/∼oferpas

DT-MRI states the relation between the signal attenuation, $E$, and the diffusion tensor, $D$, as follows [4, 5, 6]:

$$E(q_k) = \frac{A(q_k)}{A(0)} = \exp(-bq_k^T D q_k) \, , \qquad (1)$$

where $A(q_k)$ is the DWI for the k'th applied diffusion gradient direction $q_k$. The notation $A(0)$ is for the non weighted image and $b$ is a constant reflecting the experimental diffusion weighting [2]. $D$ is a second order tensor, *i.e.*, a $3 \times 3$ positive semidefinite matrix, that requires at least 6 DWIs from different non-collinear applied gradient directions to uniquely determine it. The symmetric diffusion tensor has a spectral decomposition for three eigenvectors $U^a$ and three positive eigenvalues $\lambda^a$. The relation between the eigenvalues determines the diffusion anisotropy using measures such as Fractional Anisotropy (FA) [5]:

$$FA = \sqrt{\frac{3((\lambda_1 - \langle D \rangle)^2 + (\lambda_2 - \langle D \rangle)^2 + (\lambda_3 - \langle D \rangle)^2)}{2(\lambda_1^2 + \lambda_2^2 + \lambda_3^2)}} \, , \qquad (2)$$

where $\langle D \rangle = (\lambda_1 + \lambda_2 + \lambda_3)/3$. FA is relatively high in neuronal fiber bundles (white matter), where the cylindrical geometry of fibers causes the diffusion perpendicular to the fibers be much smaller than parallel to them. Other brain tissues, such as gray matter and Cerebro-Spinal Fluid (CSF), are less confined with diffusion direction and exhibit isotropic diffusion. In cases of partial volume where neuronal fibers reside other tissue type in the same voxel, or present complex architecture, the diffusion has no longer a single pronounced orientation and therefore the FA value of the fitted tensor is decreased. The decreased FA values causes errors in segmentation and in any proceeding fiber analysis.

In this paper we focus on the case where partial volume occurs when fiber bundles are infiltrated with edema. Edema might occur in response to brain trauma, or surrounding a tumor. The brain tissue accumulate water which creates pressure and might change the fiber architecture, or infiltrate it. Since the edema consists mostly of relatively free diffusing water molecules, the diffusion attenuation increases and the anisotropy decreases. We chose to reduce the effect of edema by changing the diffusion model to a dual compartment model, assuming an isotropic compartment added to a tensor compartment.

## 2 Theory

The method we offer is based on the dual compartment model which was already demonstrated as able to reduce CSF contamination [7], where it required a large number of diffusion measurement with different diffusion times. Here we require the conventional DT-MRI data of only six diffusion measurement, and apply it on the edema case.

### 2.1 The Dual Compartment Model

The dual compartment model is described as follows:

$$E(q_k) = f \exp(-bq_k^T D_1 q_k) + (1 - f) \exp(-bD_2) \, . \qquad (3)$$

The diffusion tensor for the tensor compartment is denoted by $D_1$, and the diffusion coefficient of the isotropic water compartment is denoted by $D_2$. The compartments have relative volume of $f$ and $1 - f$. Finding the best fitting parameters $D_1$, $D_2$ and $f$ is highly ill-posed, especially in the case of six measurement, where for any arbitrarily chosen isotropic compartment there could be found a tensor compartment which exactly fits the data.

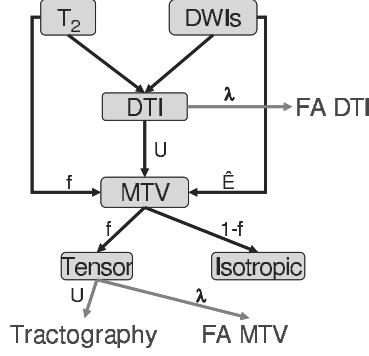

Figure 1: The initialization scheme. In addition to the DWI data, MTV uses the $T_2$ image to initialize $f$. The initial orientation for the tensor compartment are those that DT-MRI calculated.

## 2.2 The Variational Framework

In order to stabilize the fitting process we chose to use the Multiple Tensor Variational (MTV) framework [8] which was previously used to resolve partial volume caused by complex fiber architecture [9], and to reduce CSF contamination in cases of hydrocephalus [10]. We note that the dual compartment model is a special case of the more general multiple tensor model, where the number of the compartments is restricted to 2 and one of the compartments is restricted to equal eigenvalues (isotropy). Therefore the MTV framework adapted for separation of fiber compartments from edema is composed of the following functional, whose minima should provide the wanted diffusion parameters:

$$S(f, D_1, D_2) = \int_\Omega \left[ \alpha \sum_{k=1}^{d} (E(q_k) - \hat{E}(q_k))^2 + \phi(|\nabla U_i^1|) \right] d\Omega . \tag{4}$$

The notation $\hat{E}$ is for the observed diffusion signal attenuation and $E$ is calculated using (3) for $d$ different acquisition directions. $\Omega$ is the image domain with $3D$ axis $(x, y, z)$, $|\nabla I| = \sqrt{(\frac{\partial I}{\partial x})^2 + (\frac{\partial I}{\partial y})^2 + (\frac{\partial I}{\partial z})^2}$ is defined as the vector gradient norm. The notation $U_i^1$ stands for the principal eigenvector of the i'th diffusion tensor. The fixed parameters $\alpha$ is set to keep the solution closer to the observed diffusion signal. The function $\phi$ is a diffusion flow function, which controls the regularization behavior. Here we chose to use $\phi_i(s) = \sqrt{1 + \frac{s^2}{K_i^2}}$ which lead to anisotropic diffusion-like flow while preserving discontinuities [11]. The regularized fitting allows the identification of smoothed fiber compartments and reduces noise. The minimum of (4) solves the Euler-Lagrange equations, and can be found by the gradient descent scheme.

## 2.3 Initialization Scheme

Since the functional space is highly irregular (not enough measurements), the minimization process requires initial guess (figure 1), which is as close as possible to the global minimum. In order to apriori estimate the relative volume of the isotropic compartment we used a normalized diffusion non-weighted image, where high contrast correlates to larger fluid volume. In order to apriori estimate the parameters of $D_1$ we used the result of conventional DT-MRI fitting on the original data. The DT-MRI results were spectrally decomposed and the eigenvectors were used as initial guess for the eigenvectors of $D_1$. The initial guess for

the eigenvalues of $D_1$ were set to $\lambda_1 = 1.5$, $\lambda_2 = \lambda_3 = 0.4$.

## 3  methods

We demonstrate how partial volume of neuronal fiber and edema can be reduced by applying the modified MTV framework on a brain slice taken from a patient with sever edema surrounding a brain tumor. MRI was performed on a $1.5T$ MRI scanner (GE, Milwaukee). DT-MRI experiments were performed using a diffusion-weighted spin-echo echo-planar-imaging (DWI-EPI) pulse sequence. The experimental parameters were as follows: $TR/TE = 10000/98ms$, $\Delta/\delta = 31/25ms$, $b = 1000s/mm2$ with six diffusion gradient directions. $48$ slices with thickness of $3mm$ and no gap were acquired covering the whole brain with FOV of $240mm2$ and matrix of $128x128$. Number of averages was $4$, and the total experimental time was about $6$ minutes. Head movement and image distortions were corrected using a mutual information based registration algorithm [12]. The corrected DWIs were fitted to the dual compartment model via the modified MTV framework, then the isotropic compartment was omitted. FA was calculated for the remaining tensor for which FA higher than $0.25$ was considered as white matter. We compared these results to single component DT-MRI with no regularization, which was also used for initialization of the MTV fitting.

## 4  Results and Discussion

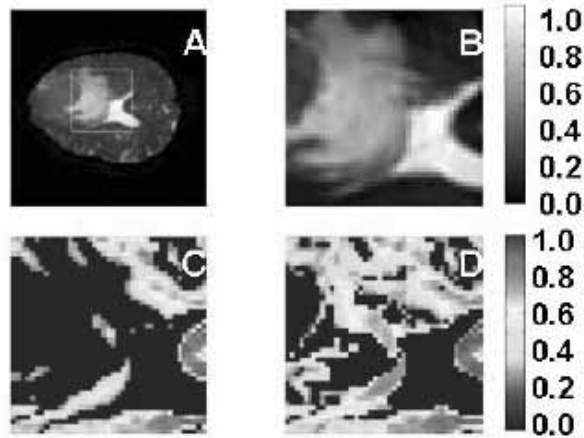

Figure 2: A single slice of a patient with edema. (A) a non diffusion weighted image with ROI marked. Showing the tumor in black surrounded by sever edema which appear bright. (B) Normalized $T_2$ of the ROI, used for $f$ initialization. (C) FA map from DT-MRI (threshold of FA$> 0.25$). Large parts of the corpus callosum are obscured. (D) FA map of $D_1$ from MTV (thresholds f$> 0.35$, FA$> 0.25$). A much larger part of the corpus callosum is revealed

Figure (2) shows the Edema case, where DTI was unable to delineate large parts of the corpus callosum. Since the corpus callosum is one of the largest fiber bundles in the brain

it was highly unlikely that the fibers were disconnected or disappeared. The expected FA should have been on the same order as on the opposite side of the brain, where the corpus callosum shows high FA values. Applying the MTV on the slice and mapping the FA value of the tensor compartment reveals considerably much more pixels of higher FA in the area of the corpus callosum. In general the FA values of most pixels were increased, which was predicted, since by removing any size of a sphere (isotropic compartment) we should be left with a shape which is less spherical, and therefore with increased FA. The benefit of using the MTV framework over an overall reduce of FA threshold in recognizing neuronal fiber voxels is that the amount of FA increase is not uniform in all tissue types. In areas where the partial volume was not big due to the edema, the increase was much lower than in areas contaminated with edema. This keeps the nice contrast reflected by FA values between neuronal fibers and other tissue types. Reducing the FA threshold on original DT-MRI results would cause a less clear separation between the fiber bundles and other tissue types. This tool could be used for fiber tracking in the vicinity of brain tumors, or with stroke, where edema contaminates the fibers and prevents fiber delineation with the conventional DT-MRI.

## 5   Conclusions

We show that by modifying the MTV framework to fit the dual compartment model we can reduce the contamination of edema, and delineate much larger fiber bundle areas. By using the MTV framework we stabilize the fitting process, and also include some biological constraints, such as the piece-wise smoothness nature of neuronal fibers in the brain. There is no doubt that using a much larger number of diffusion measurements should increase the stabilization of the process, and will increase its accuracy. However, more measurement require much more scan time, which might not be available in some cases. The variational framework is a powerful tool for the modeling and regularization of various mappings. It is applied, with great success, to scalar and vector fields in image processing and computer vision. Recently it has been generalized to deal with tensor fields which are of great interest to brain research via the analysis of DWIs and DT-MRI. We show that the more realistic model of multi-compartment voxels conjugated with the variational framework provides much improved results.

### Acknowledgments

We acknowledge the support of the Edersheim - Levi - Gitter Institute for Functional Human Brain Mapping of Tel-Aviv Sourasky Medical Center and Tel-Aviv University, the Adams super-center for brain research of Tel-Aviv University, the Israel Academy of Sciences, Israel Ministry of Science, and the Tel-Aviv University research fund.

## References

[1] E Stejskal and JE Tanner. Spin diffusion measurements: Spin echoes in the presence of a time-dependant field gradient. *J. Chem. Phys.*, 42:288–292, 1965.

[2] D. Le-Bihan, J.-F. Mangin, C. Poupon, C.A. Clark, S. Pappata, N. Molko, and H. Chabriat. Diffusion tensor imaging: concepts and applications. *Journal of Magnetic Resonance Imaging*, 13:534–546, 2001.

[3] S. Mori and P.C. van Zijl. Fiber tracking: principles and strategies - a technical review. *NMR Biomed.*, 15:468–480, 2002.

[4] P.J. Basser, J. Mattiello, and D. Le-Bihan. MR diffusion tensor spectroscopy and imaging. *Biophysical Journal*, 66:259–267, 1994.

[5] P.J. Basser and C. Pierpaoli. Microstructural and physiological features of tissues elucidated by quantitative-diffusion-tensor MRI. *Journal of Magnetic Resonance*, 111(3):209–219, June 1996.

[6] C. Pierpaoli, P. Jezzard, P.J. Basser, A. Barnett, and G. Di-Chiro. Diffusion tensor MR imaging of human brain. *Radiology*, 201:637–648, 1996.

[7] C. Pierpaoli and D. K. Jones. Removing CSF contamination in brain DT-MRIs by using a two-compartment tensor model. In *Proc. International Society for Magnetic Resonance in Medicine 12th Scientific meeting ISMRM04*, page 1215, Kyoto, Japan, 2004.

[8] O. Pasternak, N. Sochen, and Y. Assaf. Variational regularization of multiple diffusion tensor fields. In J. Weickert and H. Hagen, editors, *Visualization and Processing of Tensor Fields*. Springer, Berlin, 2005.

[9] O. Pasternak, N. Sochen, and Y. Assaf. Separation of white matter fascicles from diffusion MRI using $\phi$-functional regularization. In *Proceedings of 12th Annual Meeting of the ISMRM*, page 1227, 2004.

[10] O. Pasternak, N. Sochen, and Y. Assaf. CSF partial volume reduction in hydrocephalus using a variational framework. In *Proceedings of 13th Annual Meeting of the ISMRM*, page 1100, 2005.

[11] G. Aubert and P. Kornprobst. *Mathematical Problems in Image Processing: Partial Differential Equations and the Calculus of Variations*, volume 147 of *Applied Mathematical Sciences*. Springer-Verlag, 2002.

[12] G.K. Rohde, A.S. Barnett, P.J. Basser, S. Marenco, and C. Pierpaoli. Comprehensive approach for correction of motion and distortion in diffusion-weighted MRI. *Magnetic Resonance in Medicine*, 51:103–114, 2004.
